# Learning from Dyadic Data

**Thomas Hofmann\*, Jan Puzicha+, Michael I. Jordan\***

\* Center for Biological and Computational Learning, M.I.T
Cambridge, MA, {hofmann, jordan}@ai.mit.edu
+ Institut für Informatik III, Universität Bonn, Germany, jan@cs.uni-bonn.de

## Abstract

*Dyadic data* refers to a domain with two finite sets of objects in which observations are made for *dyads*, i.e., pairs with one element from either set. This type of data arises naturally in many application ranging from computational linguistics and information retrieval to preference analysis and computer vision. In this paper, we present a systematic, domain-independent framework of learning from dyadic data by statistical *mixture models*. Our approach covers different models with flat and hierarchical latent class structures. We propose an *annealed* version of the standard EM algorithm for model fitting which is empirically evaluated on a variety of data sets from different domains.

## 1 Introduction

Over the past decade *learning from data* has become a highly active field of research distributed over many disciplines like pattern recognition, neural computation, statistics, machine learning, and data mining. Most domain-independent learning architectures as well as the underlying theories of learning have been focusing on a feature-based data representation by vectors in an Euclidean space. For this restricted case substantial progress has been achieved. However, a variety of important problems does not fit into this setting and far less advances have been made for data types based on different representations.

In this paper, we will present a general framework for unsupervised learning from *dyadic data*. The notion *dyadic* refers to a domain with two (abstract) sets of objects, $\mathcal{X} = \{x_1, \ldots, x_N\}$ and $\mathcal{Y} = \{y_1, \ldots, y_M\}$ in which observations $\mathcal{S}$ are made for dyads $(x_i, y_k)$. In the simplest case – on which we focus – an elementary observation consists just of $(x_i, y_k)$ itself, i.e., a *co-occurrence* of $x_i$ and $y_k$, while other cases may also provide a scalar value $w_{ik}$ (strength of preference or association). Some exemplary application areas are: (i) *Computational linguistics* with the corpus-based statistical analysis of word co-occurrences with applications in language modeling, word clustering, word sense disambiguation, and thesaurus construction. (ii) *Text-based information retrieval*, where $\mathcal{X}$ may correspond to a document collection, $\mathcal{Y}$

to keywords, and $(x_i, y_k)$ would represent the occurrence of a term $y_k$ in a document $x_i$. (iii) *Modeling of preference and consumption behavior* by identifying $\mathcal{X}$ with individuals and $\mathcal{Y}$ with objects or stimuli as in *collaborative filtering*. (iv) *Computer vision*, in particular in the context of image segmentation, where $\mathcal{X}$ corresponds to image locations, $\mathcal{Y}$ to discretized or categorical feature values, and a dyad $(x_i, y_k)$ represents a feature $y_k$ observed at a particular location $x_i$.

## 2   Mixture Models for Dyadic Data

Across different domains there are at least two tasks which play a fundamental role in unsupervised learning from dyadic data: (i) probabilistic modeling, i.e., learning a joint or conditional probability model over $\mathcal{X} \times \mathcal{Y}$, and (ii) structure discovery, e.g., identifying clusters and data hierarchies. The key problem in probabilistic modeling is the *data sparseness*: How can probabilities for rarely observed or even unobserved co-occurrences be reliably estimated? As an answer we propose a model-based approach and formulate *latent class* or *mixture models*. The latter have the further advantage to offer a unifying method for probabilistic modeling and structure discovery. There are at least three (four, if both variants in (ii) are counted) different ways of defining latent class models:

i. The most direct way is to introduce an (unobserved) mapping $c : \mathcal{X} \times \mathcal{Y} \rightarrow \{c_1, \ldots, c_K\}$ that partitions $\mathcal{X} \times \mathcal{Y}$ into $K$ classes. This type of model is called *aspect-based* and the pre-image $c^{-1}(c_\alpha)$ is referred to as an *aspect*.

ii. Alternatively, a class can be defined as a subset of *one* of the spaces $\mathcal{X}$ (or $\mathcal{Y}$ by symmetry, yielding a different model), i.e., $c : \mathcal{X} \rightarrow \{c_1, \ldots, c_K\}$ which induces a unique partitioning on $\mathcal{X} \times \mathcal{Y}$ by $c(x_i, y_k) \equiv c(x_i)$. This model is referred to as *one-sided clustering* and $c^{-1}(c_\alpha) \subseteq \mathcal{X}$ is called a *cluster*.

iii. If latent classes are defined for both sets, $c : \mathcal{X} \rightarrow \{c_1^x, \ldots, c_K^x\}$ and $c : \mathcal{Y} \rightarrow \{c_1^y, \ldots, c_L^y\}$, respectively, this induces a mapping $c$ which is a $K \cdot L$-partitioning of $\mathcal{X} \times \mathcal{Y}$. This model is called *two-sided clustering*.

### 2.1   Aspect Model for Dyadic Data

In order to specify an aspect model we make the assumption that all co-occurrences in the sample set $\mathcal{S}$ are i.i.d. and that $x_i$ and $y_k$ are conditionally independent given the class. With parameters $P(x_i|c_\alpha)$, $P(y_k|c_\alpha)$ for the class-conditional distributions and prior probabilities $P(c_\alpha)$ the complete data probability can be written as

$$P(\mathcal{S}, c) = \prod_{i,k} [P(c_{ik})P(x_i|c_{ik})P(y_k|c_{ik})]^{n(x_i, y_k)}, \tag{1}$$

where $n(x_i, y_k)$ are the empirical counts for dyads in $\mathcal{S}$ and $c_{ik} \equiv c(x_i, y_k)$. By summing over the latent variables $c$ the usual mixture formulation is obtained

$$P(\mathcal{S}) = \prod_{i,k} P(x_i, y_k)^{n(x_i, y_k)}, \quad \text{where} \quad P(x_i, y_k) = \sum_\alpha P(c_\alpha)P(x_i|c_\alpha)P(y_k|c_\alpha). \tag{2}$$

Following the standard *Expectation Maximization* approach for maximum likelihood estimation [Dempster *et al.*, 1977], the E-step equations for the class posterior probabilities are given by[1]

$$P\{c_{ik} = c_\alpha\} \propto P(c_\alpha)P(x_i|c_\alpha)P(y_j|c_\alpha). \tag{3}$$

| $P(c_\alpha)$ | #1, 0.004 | #2, 0.005 | #3, 0.008 | #4, 0.002 | #5, 0.040 | #6, 0.011 | #7, 0.029 | #8, 0.007 |
|---|---|---|---|---|---|---|---|---|
| maximal $P(x_i\|c_\alpha)$ | two 0.18<br>seven 0.10<br>three 0.10<br>four 0.06<br>five 0.06 | went 0.10<br>go 0.08<br>come 0.04<br>came 0.04<br>brought 0.03 | have 0.38<br>hath 0.22<br>had 0.11<br>hast 0.09<br>be 0.02 | shalt 0.18<br>hast 0.08<br>wilt 0.08<br>art 0.07<br>if 0.05 | the 0.95<br>his 0.006<br>my 0.005<br>our 0.003<br>thy 0.003 | he 0.51<br>god 0.08<br>lord 0.05<br>and 0.04<br>who 0.03 | <.> 0.52<br><:> 0.16<br><,> 0.14<br><;> 0.07<br><?> 0.04 | thee 0.04<br>me 0.03<br>him 0.03<br>it 0.02<br>you 0.02 |
| maximal $P(y_k\|c_\alpha)$ | years 0.11<br>thousand 0.1<br>hundred 0.1<br>days 0.07<br>cubits 0.05 | up 0.40<br>down 0.17<br>forth 0.15<br>out 0.09<br>in 0.01 | not 0.04<br>done 0.04<br>given 0.03<br>made 0.03<br>been 0.03 | thou 0.85<br>not 0.01<br>also 0.004<br>indeed 0.003<br>anoint 0.003 | lord 0.09<br>children 0.02<br>son 0.02<br>land 0.02<br>people 0.02 | hath 0.14<br>shall 0.07<br>said 0.05<br>is 0.04<br>was 0.04 | and 0.33<br>for 0.08<br>but 0.07<br>then 0.05<br>so 0.02 | <?> 0.27<br><,> 0.23<br><.> 0.12<br><:> 0.06<br><:> 0.04 |

Figure 1: Some aspects of the *Bible* (bigrams).

It is straightforward to derive the M-step re-estimation formulae

$$P(c_\alpha) \propto \sum_{i,k} n(x_i, y_k) P\{c_{ik} = c_\alpha\}, \quad P(x_i|c_\alpha) \propto \sum_k n(x_i, y_k) P\{c_{ik} = c_\alpha\}, \qquad (4)$$

and an analogous equation for $P(y_k|c_\alpha)$. By re-parameterization the aspect model can also be characterized by a cross-entropy criterion. Moreover, formal equivalence to the *aggregate Markov model*, independently proposed for language modeling in [Saul, Pereira, 1997], has been established (cf. [Hofmann, Puzicha, 1998] for details).

## 2.2 One-Sided Clustering Model

The complete data model proposed for the one-sided clustering model is

$$P(\mathcal{S}, c) = P(c)P(\mathcal{S}|c) = \left( \prod_i P(c(x_i)) \right) \left( \prod_{i,k} [P(x_i)P(y_k|c(x_i))]^{n(x_i,y_k)} \right), \qquad (5)$$

where we have made the assumption that observations $(x_i, y_k)$ for a particular $x_i$ are conditionally independent given $c(x_i)$. This effectively defines the mixture

$$P(\mathcal{S}) = \prod_i P(\mathcal{S}_i), \quad P(\mathcal{S}_i) = \sum_\alpha P(c_\alpha) \prod_k [P(x_i)P(y_k|c_\alpha)]^{n(x_i,y_k)}, \qquad (6)$$

where $\mathcal{S}_i$ are all observations involving $x_i$. Notice that co-occurrences in $\mathcal{S}_i$ are not independent (as they are in the aspect model), but get coupled by the (shared) latent variable $c(x_i)$. As before, it is straightforward to derive an EM algorithm with update equations

$$P\{c(x_i) = c_\alpha\} \propto P(c_\alpha) \prod_k P(y_k|c_\alpha)^{n(x_i,y_k)}, \quad P(y_k|c_\alpha) \propto \sum_i n(x_i, y_k) P\{c(x_i) = c_\alpha\} \qquad (7)$$

and $P(c_\alpha) \propto \sum_i P\{c(x_i) = c_\alpha\}$, $P(x_i) \propto \sum_j n(x_i, y_j)$. The one-sided clustering model is similar to the distributional clustering model [Pereira *et al.*, 1993], however, there are two important differences: (i) the number of likelihood contributions in (7) scales with the number of observations – a fact which follows from Bayes' rule – and (ii) mixing proportions are missing in the original distributional clustering model. The one-sided clustering model corresponds to an unsupervised version of the naive Bayes' classifier, if we interpret $\mathcal{Y}$ as a feature space for objects $x_i \in \mathcal{X}$. There are also ways to weaken the conditional independence assumption, e.g., by utilizing a mixture of tree dependency models [Meila, Jordan, 1998].

## 2.3 Two-Sided Clustering Model

The latent variable structure of the two-sided clustering model significantly reduces the degrees of freedom in the specification of the class conditional distribution. We

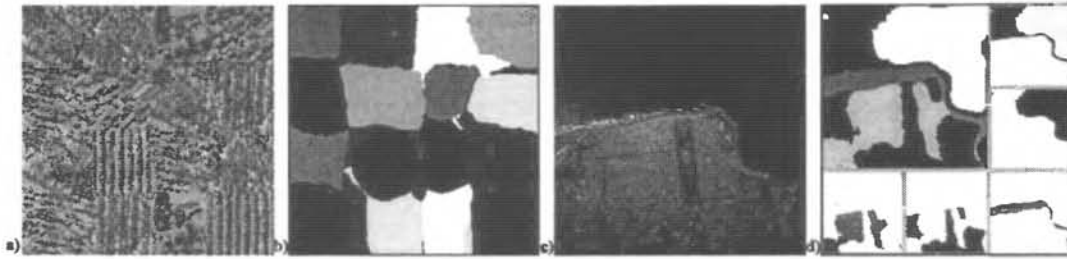

Figure 2: Exemplary segmentation results on *Aerial* by one–sided clustering.

propose the following complete data model

$$P(\mathcal{S}, c) = \prod_{i,k} P(c(x_i))P(c(y_k)) \left[ P(x_i)P(y_k)\pi_{c(x_i),c(y_k)} \right]^{n(x_i,y_k)} \tag{8}$$

where $\pi_{c_\alpha^x,c_\gamma^y}$ are cluster association parameters. In this model the latent variables in the $\mathcal{X}$ and $\mathcal{Y}$ space are coupled by the $\pi$–parameters. Therefore, there exists no simple mixture model representation for $P(\mathcal{S})$. Skipping some of the technical details (cf. [Hofmann, Puzicha, 1998]) we obtain $P(x_i) \propto \sum_k n(x_i, y_k)$, $P(y_k) \propto \sum_i n(x_i, y_k)$ and the M-step equations

$$\pi_{c_\alpha^x,c_\gamma^y} = \frac{\sum_{i,k} n(x_i, y_k)P\{c(x_i) = c_\alpha^x \wedge c(y_k) = c_\gamma^y\}}{[\sum_i P\{c(x_i) = c_\alpha^x\} \sum_k n(x_i, y_k)][\sum_k P\{c(y_k) = c_\gamma^y\} \sum_i n(x_i, y_k)]} \tag{9}$$

as well as $P(c_\alpha^x) = \sum_i P\{c(x_i) = c_\alpha^x\}$ and $P(c_\gamma^y) = \sum_k P\{c(x_k) = c_\gamma^y\}$. To preserve tractability for the remaining problem of computing the posterior probabilities in the E-step, we apply a factorial approximation (*mean field* approximation), i.e., $P\{c(x_i) = c_\alpha^x \wedge c(y_k) = c_\gamma^y\} \approx P\{c(x_i) = c_\alpha^x\}P\{c(y_k) = c_\gamma^y\}$. This results in the following coupled approximation equations for the marginal posterior probabilities

$$P\{c(x_i) = c_\alpha^x\} \propto P(c_\alpha^x) \exp \left[ \sum_k n(x_i, y_k) \sum_\gamma P\{c(y_k) = c_\gamma^y\} \log \pi_{c_\alpha^x,c_\gamma^y} \right] \tag{10}$$

and a similar equation for $P\{c(y_k) = c_\gamma^y\}$. The resulting approximate EM algorithm performs updates according to the sequence ($c^x$–post., $\pi$, $c^y$–post., $\pi$). Intuitively the (probabilistic) clustering in one set is optimized in alternation for a given clustering in the other space and vice versa. The two–sided clustering model can also be shown to maximize a mutual information criterion [Hofmann, Puzicha, 1998].

## 2.4 Discussion: Aspects and Clusters

To better understand the differences of the presented models it is elucidating to systematically compare the conditional probabilities $P(c_\alpha|x_i)$ and $P(c_\alpha|y_k)$:

| | Aspect Model | One-sided $\mathcal{X}$ Clustering | One-sided $\mathcal{Y}$ Clustering | Two-sided Clustering |
|---|---|---|---|---|
| $P(c_\alpha\|x_i)$ | $\frac{P(x_i\|c_\alpha)P(c_\alpha)}{P(x_i)}$ | $P\{c(x_i) = c_\alpha\}$ | $\frac{P(x_i\|c_\alpha)P(c_\alpha)}{P(x_i)}$ | $P\{c(x_i) = c_\alpha^x\}$ |
| $P(c_\alpha\|y_k)$ | $\frac{P(y_k\|c_\alpha)P(c_\alpha)}{P(y_k)}$ | $\frac{P(y_k\|c_\alpha)P(c_\alpha)}{P(y_k)}$ | $P\{c(y_k) = c_\alpha\}$ | $P\{c(y_k) = c_\alpha^y\}$ |

As can be seen from the above table, probabilities $P(c_\alpha|x_i)$ and $P(c_\alpha|y_k)$ correspond to posterior probabilities of latent variables if clusters are defined in the $\mathcal{X}$– and $\mathcal{Y}$–space, respectively. Otherwise, they are computed from model *parameters*. This is a crucial difference as, for example, the posterior probabilities are approaching

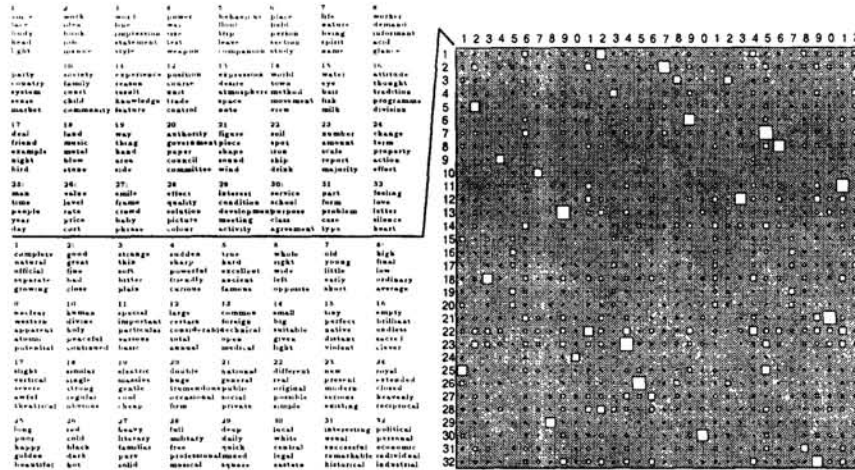

Figure 3: Two–sided clustering of LOB: $\pi$ matrix and most probable words.

Boolean values in the infinite data limit and $P(y_k|x_i) = \sum_\alpha P\{c(x_i) = c_\alpha\} P(y_k|c_\alpha)$ are converging to one of the class-conditional distributions. Yet, in the aspect model $P(y_k|x_i) = \sum_\alpha P(c_\alpha|x_i)P(y_k|c_\alpha)$ and $P(c_\alpha|x_i) \propto P(c_\alpha)P(x_i|c_\alpha)$ are typically not peaking more sharply with an increasing number of observations. In the aspect model, conditionals $P(y_k|x_i)$ are inherently a weighted sum of the 'prototypical' distributions $P(y_k|c_\alpha)$. Cluster models in turn ultimately look for the 'best' class-conditional and weights are only indirectly induced by the posterior uncertainty.

## 3   The Cluster-Abstraction Model

The models discussed in Section 2 all define a non-hierarchical, 'flat' latent class structure. However, for structure discovery it is important to find hierarchical data organizations. There are well-known architectures like the *Hierarchical Mixtures of Experts* [Jordan, Jacobs, 1994] which fit hierarchical models. Yet, in the case of dyadic data there is an alternative possibility to define a hierarchical model. The *Cluster-Abstraction Model* (CAM) is a clustering model (e.g., in $\mathcal{X}$) where the conditionals $P(y_k|c_\alpha)$ are itself $x_i$–specific aspect mixtures, $P(y_k|c_\alpha, x_i) = \sum_\nu P(y_k|a_\nu)P(a_\nu|c_\alpha, x_i)$ with a latent aspect mapping $a$. To obtain a hierarchical organization, clusters $c_\alpha$ are identified with the terminal nodes of a hierarchy (e.g., a complete binary tree) and aspects $a_\nu$ with inner *and* terminal nodes. As a compatibility constraint it is imposed that $P(a_\nu|c_\alpha, x_i) = 0$ whenever the node corresponding to $a_\nu$ is not on the path to the terminal node $c_\alpha$. Intuitively, conditioned on a 'horizontal' clustering $c$ all observations $(x_i, y_k) \in \mathcal{S}_i$ for a particular $x_i$ have to be generated from one of the 'vertical' abstraction levels on the path to $c(x_i)$. Since different clusters share aspects according to their topological relation, this favors a meaningful hierarchical organization of clusters. Moreover, aspects at inner nodes do not simply represent averages over clusters in their subtree as they are forced to explicitly represent what is *common* to all subsequent clusters.

Skipping the technical details, the E-step is given by

$$P\{a(x_i, y_k) = a_\nu | c(x_i) = c_\alpha\} \propto P(a_\nu|c_\alpha, x_i)P(y_k|a_\nu) \tag{11}$$

$$P\{c(x_i) = c_\alpha\} \propto P(c_\alpha) \prod_k \sum_\nu [P(a_\nu|c_\alpha, x_i)P(y_k|a_\nu)]^{n(x_i, y_k)} \tag{12}$$

and the M-step formulae are $P(y_k|a_\nu) \propto \sum_i P\{a(x_i, y_k) = a_\nu\}n(x_i, y_k)$, $P(c_\alpha) \propto \sum_i P\{c(x_i) = c_\alpha\}$, and $P(a_\nu|c_\alpha, x_i) \propto \sum_k P\{a(x_i, y_k) = a_\nu|c(x_i) = c_\alpha\}n(x_i, y_k)$.

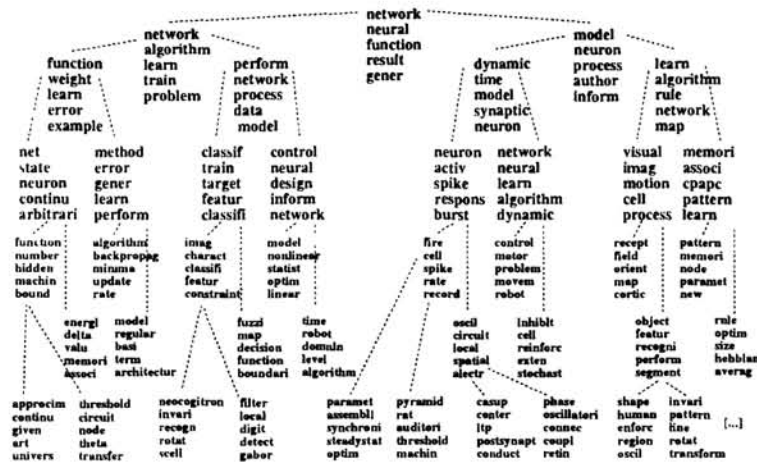

Figure 4: Parts of the top levels of a hierarchical clustering solution for the *Neural* document collection, aspects are represented by their 5 most probable word stems.

## 4   Annealed Expectation Maximization

Annealed EM is a generalization of EM based on the idea of *deterministic anneal-ing* [Rose *et al.*, 1990] that has been successfully applied as a heuristic optimization technique to many clustering and mixture problems. Annealing reduces the sensitivity to local maxima, but, even more importantly in this context, it may also improve the generalization performance compared to maximum likelihood estimation.[2] The key idea in annealed EM is to introduce an (inverse temperature) parameter $\beta$, and to replace the negative (averaged) complete data log-likelihood by a substitute known as the *free energy* (both are in fact equivalent at $\beta = 1$). This effectively results in a simple modification of the E-step by taking the likelihood contribution in Bayes' rule to the power of $\beta$. In order to determine the optimal value for $\beta$ we used an additional validation set in a cross validation procedure.

## 5   Results and Conclusions

In our experiments we have utilized the following real-world data sets: (i) *Cranfield*: a standard test collection from information retrieval ($N = 1400$, $M = 4898$), (ii) *Penn*: adjective-noun co-occurrences from the Penn Treebank corpus ($N = 6931$, $M = 4995$) and the LOB corpus ($N = 5448$, $M = 6052$), (iii) *Neural*: a document collection with abstracts of journal papers on neural networks ($N = 1278$, $M = 6065$), (iv) *Bible*: word bigrams from the bible edition of the Gutenberg project ($N = M = 12858$), (v) *Aerial*: Textured aerial images for segmentation ($N = 128\times128$, $M = 192$).

In Fig. 1 we have visualized an aspect model fitted to the *Bible* bigram data. Notice that although $\mathcal{X} = \mathcal{Y}$ the role of the preceding and the subsequent words in bigrams is quite different. Segmentation results obtained on *Aerial* applying the one-sided clustering model are depicted in Fig. 2. A multi-scale Gabor filter bank (3 octaves, 4 orientations) was utilized as an image representation (cf. [Hofmann *et al.*, 1998]). In Fig. 3 a two-sided clustering solution of LOB is shown. Fig. 4 shows the top levels of the hierarchy found by the Cluster-Abstraction Model in *Neural*. The inner node distributions provide resolution-specific descriptors for the documents in the corresponding subtree which can be utilized, e.g., in interactive browsing for information retrieval. Fig. 5 shows typical test set perplexity curves of the

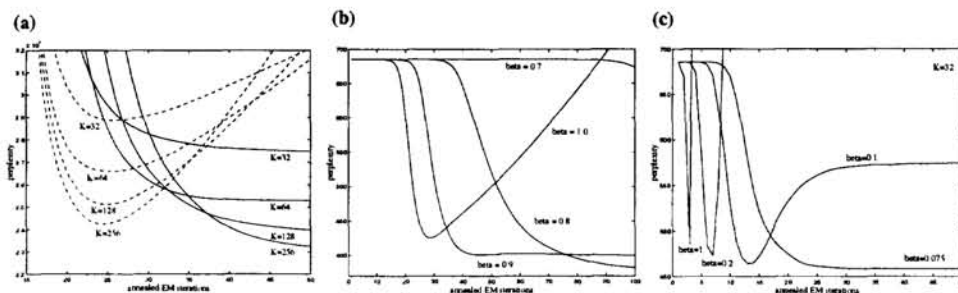

Figure 5: Perplexity curves for annealed EM (aspect (a), (b) and one-sided clustering model (c)) on the *Bible* and *Cran* data.

| K | Aspect $\beta$ | Aspect $\mathcal{P}$ | $\mathcal{X}$-cluster $\beta$ | $\mathcal{X}$-cluster $\mathcal{P}$ | CAM $\beta$ | CAM $\mathcal{P}$ | $\mathcal{X}/\mathcal{Y}$-cluster $\beta$ | $\mathcal{X}/\mathcal{Y}$-cluster $\mathcal{P}$ | Aspect $\beta$ | Aspect $\mathcal{P}$ | $\mathcal{X}$-cluster $\beta$ | $\mathcal{X}$-cluster $\mathcal{P}$ | CAM $\beta$ | CAM $\mathcal{P}$ | $\mathcal{X}/\mathcal{Y}$-cluster $\beta$ | $\mathcal{X}/\mathcal{Y}$-cluster $\mathcal{P}$ |
|---|---|---|---|---|---|---|---|---|---|---|---|---|---|---|---|---|
| | Cran | | | | | | | | Penn | | | | | | | |
| 1 | - | 685 | - | - | - | - | - | - | - | 639 | - | - | - | - | - | - |
| 8 | 0.88 | 482 | 0.09 | 527 | 0.18 | 511 | 0.67 | 615 | 0.73 | 312 | 0.08 | 352 | 0.13 | 322 | 0.55 | 394 |
| 16 | 0.72 | 255 | 0.07 | 302 | 0.10 | 268 | 0.51 | 335 | 0.72 | 255 | 0.07 | 302 | 0.10 | 268 | 0.51 | 335 |
| 32 | 0.83 | 386 | 0.07 | **452** | 0.12 | 438 | 0.53 | 506 | 0.71 | 205 | 0.07 | 254 | 0.08 | 226 | 0.46 | 286 |
| 64 | 0.79 | 360 | 0.06 | 527 | 0.11 | 422 | 0.48 | 477 | 0.69 | 182 | 0.07 | **223** | 0.07 | 204 | 0.44 | 272 |
| 128 | 0.78 | **353** | 0.04 | 663 | 0.10 | **410** | 0.45 | **462** | 0.68 | **166** | 0.06 | 231 | 0.06 | **179** | 0.40 | **241** |

Table 1: Perplexity results for different models on the *Cran* (predicting words conditioned on documents) and *Penn* data (predicting nouns conditioned on adjectives).

annealed EM algorithm for the aspect and clustering model ($\mathcal{P} = e^{-l}$ where $l$ is the per-observation log-likelihood). At $\beta = 1$ (standard EM) overfitting is clearly visible, an effect that vanishes with decreasing $\beta$. Annealed learning performs also better than standard EM with early stopping. Tab. 1 systematically summarizes perplexity results for different models and data sets.

In *conclusion* mixture models for dyadic data have shown a broad application potential. Annealing yields a substantial improvement in generalization performance compared to standard EM, in particular for clustering models, and also outperforms a complexity control via $K$. In terms of perplexity, the aspect model has the best performance. Detailed performance studies and comparisons with other state-of-the-art techniques will appear in forthcoming papers.

## Footnotes

[1] In the case of multiple observations of dyads it has been assumed that each observation may have a different latent class. If only one latent class variable is introduced for each dyad, slightly different equations are obtained.

[2]Moreover, the tree topology for the CAM is heuristically grown via phase transitions.

# References

[Dempster *et al.*, 1977] Dempster, A.P., Laird, N.M., Rubin, D.B. (1977). Maximum likelihood from incomplete data via the EM algorithm. *J. Royal Statist. Soc. B*, **39**, 1–38.

[Hofmann, Puzicha, 1998] Hofmann, T., Puzicha, J. 1998. *Statistical models for co-occurrence data*. Tech. rept. Artifical Intelligence Laboratory Memo 1625, M.I.T.

[Hofmann *et al.*, 1998] Hofmann, T., Puzicha, J., Buhmann, J.M. (1998). Unsupervised texture segmentation in a deterministic annealing framework. *IEEE Transactions on Pattern Analysis and Machine Intelligence*, **20**(8), 803–818.

[Jordan, Jacobs, 1994] Jordan, M.I., Jacobs, R.A. (1994). Hierarchical mixtures of experts and the EM algorithm. *Neural Computation*, **6**(2), 181–214.

[Meila, Jordan, 1998] Meila, M., Jordan, M. I. 1998. Estimating Dependency Structure as a Hidden Variable. *In: Advances in Neural Information Processing Systems 10*.

[Pereira *et al.*, 1993] Pereira, F.C.N., Tishby, N.Z., Lee, L. 1993. Distributional clustering of English words. *Pages 183–190 of: Proceedings of the ACL*.

[Rose *et al.*, 1990] Rose, K., Gurewitz, E., Fox, G. (1990). Statistical mechanics and phase transitions in clustering. *Physical Review Letters*, **65**(8), 945–948.

[Saul, Pereira, 1997] Saul, L., Pereira, F. 1997. Aggregate and mixed–order Markov models for statistical language processing. *In: Proceedings of the 2nd International Conference on Empirical Methods in Natural Language Processing*.
